# Identifying Structure across Pre-partitioned Data

**Zvika Marx**
Neural Computation Center
The Hebrew University
Jerusalem, Israel, 91904

**Ido Dagan**
Department of CS
Bar-Ilan University
Ramat-Gan, Israel, 52900

**Eli Shamir**
School for CS
The Hebrew University
Jerusalem, Israel, 91904

## Abstract

We propose an information-theoretic clustering approach that incorporates a pre-known partition of the data, aiming to identify common clusters that cut across the given partition. In the standard clustering setting the formation of clusters is guided by a single source of feature information. The newly utilized pre-partition factor introduces an additional bias that counterbalances the impact of the features whenever they become correlated with this known partition. The resulting algorithmic framework was applied successfully to synthetic data, as well as to identifying text-based cross-religion correspondences.

## 1    Introduction

The standard task of feature-based data clustering deals with a single set of elements that are characterized by a unified set of features. The goal of the clustering task is to identify implicit constructs, or themes, within the clustered set, grouping together elements that are characterized similarly by the features. In recent years there has been growing interest in more complex clustering settings, in which additional information is incorporated [1], [2]. Several such extensions ([3]-[5]) are based on the *information bottleneck* (IB) framework [6], which facilitates coherent information-theoretic representation of different information types.

In a recent line of research we have investigated the *cross-dataset clustering* task [7], [8]. In this setting, some inherent a-priori partition of the clustered data to distinct subsets is given. The clustering goal it to identify corresponding (analogous) structures that cut across the different subsets, while ignoring internal structures that characterize individual subsets. To accomplish this task, those features that commonly characterize elements across the different subsets guide the clustering process, while within-subset regularities are neutralized.

In [7], we presented a distance-based hard clustering algorithm for the *coupled-clustering* problem, in which the clustered data is pre-partitioned to two subsets. In [8], our setting, generalized to pre-partitions of any number of subsets, was addressed by a heuristic extension of the probabilistic IB algorithm, yielding improved empirical results. Specifically, the algorithm in [8] was based on a

modification of the IB stable-point equation, which amplified the impact of features characterizing a formed cluster across all, or most, subsets.

This paper describes an information-theoretic framework that motivates and extends the algorithm proposed in [8]. The given pre-partitioning is represented via a probability distribution variable, which may represent "soft" pre-partitioning of the data, versus the strictly disjoint subsets assumed in the earlier cross-dataset framework. Further, we present a new functional that captures the cross-partition motivation. From the new functional, we derive a stable-point equation underlying our algorithmic framework in conjunction with the corresponding IB equation.

Our algorithm was tested empirically on synthetic data and on a real-world text-based task that aimed to identify corresponding themes across distinct religions. We have cross-clustered five sets of keywords that were extracted from topical corpora of texts about *Buddhism*, *Christianity*, *Hinduism*, *Islam* and *Judaism*. In distinction from standard clustering results, our algorithm reveals themes that are common to all religions, such as *sacred writings*, *festivals*, *narratives and myths* and *theological principles*, and avoids topical clusters that correspond to individual religions (for example, 'Christmas' and 'Easter' are clustered together with 'Ramadan' rather than with 'Church').

Finally, we have paid specific attention to the framework of clustering with side information [4]. While this approach was presented for a somewhat different mindset, it might be used directly to address clustering across pre-partitioned data. We compare the technical details of the two approaches and demonstrate empirically that clustering with side information does not seem appropriate for the kind of cross-partition tasks that we explored.

## 2    The Information Bottleneck Method

Probabilistic ("soft") data clustering outputs, for each element $x$ of the set being clustered and each cluster $c$, an assignment probability $p(c|x)$. The IB method [6] interprets probabilistic clustering as lossy data compression. The given data is represented by a random variable $X$ ranging over the clustered elements. $X$ is compressed through another random variable $C$, ranging over the clusters. Every element $x$ is characterized by conditional probability distribution $p(Y|x)$, where $Y$ is a third random variable taking the members $y$ of a given set of features as values.

The IB method formalizes the clustering task as minimizing the *IB functional*:

$$L^{(IB)} = I(C;X) - \beta I(C;Y) . \tag{1}$$

As known from information theory (Ch. 13 of [9]), minimizing the mutual information $I(C;X)$ optimizes distorted compression rate. A complementary bias to maximize $I(C;Y)$ is interpreted in [6] as articulating the level of relevance of $Y$ to the obtained clustering, inferred from the level by which $C$ can predict $Y$. $\beta$ is a free parameter counterbalancing the two biases. It is shown in [6] that $p(c|x)$ values that minimize $L^{(IB)}$ satisfy the following equation:

$$p(c|x) = \frac{1}{z(\beta,x)} p(c) e^{-\beta D_{KL}[p(Y|x)\|p(Y|c)]} , \tag{2}$$

where $D_{KL}$ stands for the Kullback-Leibler (KL) divergence, or relative entropy, between two distributions and $z(\beta,x)$ is a normalization function over $C$. Eq. (2) implies that, optimally, $x$ is assigned to $c$ in proportion to their KL distance in a feature distribution space, where the distribution $p(Y|c)$ takes the role of a

*Start at time t = 0 and iterate the following update-steps, till convergence:*

IB1:   initialize $p_t(c|x)$ randomly or arbitrarily     $(t = 0)$

$$p_t(c|x) \quad \propto \quad p_{t-1}(c)e^{-\beta D_{KL}[p(Y|x)\|p_{t-1}(Y|c)]} \qquad (t > 0)$$

IB2:   $p_t(c) \quad = \quad \sum_x p_t(c \mid x)p(x)$

IB3:   $p_t(y|c) \quad = \quad \dfrac{1}{p_t(c)}\sum_x p_t(c \mid x)p(y \mid x)p(x)$

Figure 1: The Information Bottleneck iterative algorithm (with fixed $\beta$ and $|C|$).

representative, or *centroid*, of *c*. The feature variable *Y* is hence utilized as the (exclusive) means to guide clustering, beyond the random nature of compression.

Figure 1 presents the IB iterative algorithm for a fixed value of $\beta$. The IB1 update step follows Eq. (2). The other two steps, which are derived from the IB functional as well, estimate the $p(c)$ and $p(y|c)$ values required for the next iteration. The algorithm converges to a local minimum of the IB functional. The IB setting, particularly the derivation of steps IB1 and IB3 of the algorithm, assumes that *Y* and *C* are independent given *X*, that is: $I(C;Y|X) = \sum_x p(x)I(C|x;Y|x) = 0$.

The balancing parameter $\beta$ affects the number of distinct clusters being formed in a manner that resembles (inverse) temperature in physical systems. The higher $\beta$ is (i.e., the stronger the bias to construct *C* that predicts *Y* well), more distinct clusters are required for encoding the data. For each $|C| = 2, 3, \dots$, there is a minimal $\beta$ value, enabling the formation of $|C|$ distinct clusters. Setting $\beta$ to be smaller than this critical value corresponding to the current $|C|$ would result in two or more clusters that are identical to one another. Based on this, the iterative algorithm is applied repeatedly within a gradual cooling-like (*deterministic annealing*) scheme: starting with random initialization of the $p_0(c|x)$'s, generate two clusters with the critical $\beta$ value, found empirically, for $|C| = 2$. Then, use a perturbation on the obtained two-cluster configuration to initialize the $p_0(c|x)$'s for a larger set of clusters and execute additional runs of the algorithm to identify the critical $\beta$ value for the larger $|C|$. And so on: each output configuration is used as a basis for a more granular one. The final outcome is a "soft hierarchy" of probabilistic clusters.

# 3    Cross-partition Clustering

*Cross-partition* (CP) clustering introduces a factor – a pre-given partition of the clustered data – additional to what considered in a standard clustering setting. For representing this factor we introduce the *pre-partitioning variable W*, ranging over all parts *w* of the pre-given partition. Every data element *x* is associated with *W* through a given probability distribution $p(W|x)$. Our goal is to cluster the data, so that the clusters *C* would not be correlated with *W*. We notice that *Y*, which is intended to direct the formation of clusters, might be a-priori correlated with *W*, so the formed clusters might end up being correlated with *W* as well. Our method aims at eliminating this aspect of *Y*.

## 3.1    Information Defocusing

As noted, some of the information conveyed by *Y* characterizes structures correlated with *W*, while the other part of the information characterizes the target cross-*W*

structures. We are interested in detecting the latter while filtering out the former. However, there is no direct a-priori separation between the two parts of the *Y*-mediated information. Our strategy in tackling this difficulty is: we follow in general *Y*'s directions, as the IB method does, while avoiding *Y*'s impact whenever it entails undesired inter-dependencies of *C* and *W*.

Our strategy implies conflicting biases with regard to the mutual information $I(C,Y)$: it should be maximized in order to form meaningful clusters, but be minimized as well in the specific context where *Y* entails *C–W* dependencies. Accordingly, we propose a computational procedure directed by two distinct cost-terms in tandem. The first one is the IB functional (Eq. 1), introducing the bias to maximize $I(C,Y)$. With this bias alone, *Y* might dictate (or "explain", in retrospect) substantial *C–W* dependencies, implying a low $I(C;W|Y)$ value.[1] Hence, the guideline of preventing *Y* from accounting for *C–W* dependencies is realized through an opposing bias of *maximizing* $I(C;W|Y) = \sum_y p(y) I(C|y; W|y)$. The second cost term – the *Information Defocusing* (ID) *functional* – consequently counterbalances minimization of $I(C,Y)$ against the new bias:

$$L^{(ID)} \;=\; I(C;Y) - \eta\, I(C;W|Y), \tag{3}$$

where $\eta$ is a free parameter articulating the tradeoff between the biases. The ID functional captures our goal of reducing the impact of *Y* selectively: "defocusing" a specific aspect of the information *Y* conveys: the information correlated with *W*.

In a like manner to the stable-point equation of the IB functional (Eq. 2), we derive the following stable-point equation for the ID functional:

$$p(c|y) = \frac{1}{z(\eta, y)}\, p(c) \prod_w p(y \mid c, w)^{\frac{\eta}{\eta+1} p(w)}, \tag{4}$$

where $z(\eta,y)$ is a normalization function over *C*. The derivation relies on an additional assumption, $I(C;W) = 0$, imposing the intended independence between *C* and *W* (the detailed derivation will be described elsewhere).

The intuitive interpretation of Eq. (4) is as follows: a feature *y* is to be associated with a cluster *c* in proportion to a weighted, though flattened, geometric mean of the "*W*-projected centroids" $p(y|c,w)$, priored by $p(c)$.[2] This scheme overweighs *y*'s that contribute to *c* evenly across *W*. Thus, clusters satisfying Eq. (4) are situated around centroids biased towards evenly contributing features. The higher $\eta$ is, heavier emphasis is put on suppressing disagreements between the *w*'s. For $\eta \to \infty$ a plain weighted geometric-mean scheme is obtained. The inclusion of a step derived from Eq. (4) in our algorithm (see below) facilitates convergence on a configuration with centroids dominated by features that are evenly distributed across *W*.

## 3.2    The Cross-partition Clustering Algorithm

Our proposed *cross partition* (CP) *clustering* algorithm (Fig. 2) seeks a clustering configuration that optimizes simultaneously both the IB and ID functionals,

*Start at time t = 0 and iterate the following update-steps, till convergence:*

CP1: Initialize $p_t(c|x)$ randomly or arbitrarily       $(t = 0)$

$$p_t(c|x) \;\; \propto \;\; p_{t-1}(c) e^{-\beta\, D_{KL}\left[\, p(Y|x) \| p_{t-1}(Y|c)\,\right]} \qquad (t > 0)$$

CP2: $\;\; p_t(c) \;\;\;\; = \;\;\; \sum_x p_t(c\,|\,x)\, p(x)$

CP3: $\;\; p^*_t(y|c,w) \;\; = \;\; \dfrac{1}{p_t(c)\, p(w)} \sum_x p_t(c\,|\,x)\, p(y\,|\,x)\, p(w\,|\,x)\, p(x)$

CP4: Initialize $p^*_t(c)$ randomly or arbitrarily       $(t = 0)$

$$p^*_t(c) \;\;\; = \;\;\; \sum_y p^*_{t-1}(c\,|\,y)\, p(y) \qquad (t > 0)$$

CP5: $\;\; p^*_t(c|y) \;\; \propto \;\; p^*_t(c) \prod_w p^*_t(y\,|\,c,w)^{\frac{\eta}{\eta+1} p(w)}$

CP6: $\;\; p_t(y|c) \;\;\; = \;\;\; \dfrac{p^*_t(c\,|\,y)\, p(y)}{p^*_t(c)}$

Figure 2: The cross-partition clustering iterative algorithm (with fixed $\beta$, $\eta$, and $|C|$).

thus obtaining clusters that cut across the pre-given partition $W$. To this end, the algorithm interleaves an iterative computation of the stable-point equations, and the additional estimated parameters, for both functionals. Steps CP1, CP2 and CP6 correspond to the computations related to the IB functional, while steps CP3, CP4 and CP5, which compute a separate set of parameters (denoted by an asterisk), correspond to the ID functional. Figure 3 summarizes the roles of the two functionals in the dynamics of the CP algorithm. The two components of the iterative cycle are tied together in steps CP3 and CP6, in which parameters from one set are used as input to compute a parameter of other set. The derivation of step CP3 relies on an additional assumption, namely that $C$, $Y$ and $W$ are jointly independent given $X$. This assumption, which extends to $W$ the underlying assumption of the IB setting that $C$ and $Y$ are independent given $X$, still entails the IB stable point equation. At convergence, the stable point equations for both the IB and ID functionals are satisfied, each by its own set of parameters (in steps CP1 and CP5).

The deterministic annealing scheme, which gradually increases $\beta$ over repeated runs (see Sec. 2), is applied for the CP algorithm as well with $\eta$ held fixed. For a given target number of clusters $|C|$, the algorithm empirically converges with a wide range of $\eta$ values[3].

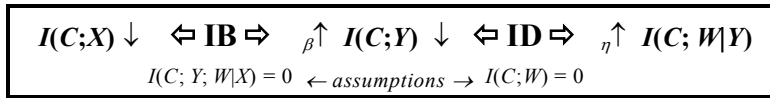

Figure 3: The interplay of the IB and the ID functionals in the CP algorithm.

# 4    Experimental Results

Our synthetic setting consisted of 75 virtual elements, evenly pre-partitioned into three 25-element parts denoted $X_1$, $X_2$ and $X_3$ (in our formalism, for each clustered element $x$, $p(w|x) = 1$ holds for either $w = 1$, 2, or 3). On top of this pre-partition, we partitioned the data twice, getting two (exhaustive) clustering configurations:

1. Target *cross-W* clustering: five clusters, each with representatives from all $X_w$'s;

2. Masking *within-w* clustering: six clusters, each consisting of roughly half the elements of either $X_1$, $X_2$ or $X_3$ with no representatives from the other $X_w$'s.

Each cluster, of both configurations, was characterized by a designated subset of features. Masking clusters were designed to be more salient than target clusters: they had more designated features (60 vs. 48 per cluster, i.e., 360 vs. 240 in total) and their elements shared higher feature-element (virtual) co-occurrence counts with those designated features (900 vs. 450 per element-feature pair). Noise (random positive integer < 200) was added to all counts associating elements with their designated features (for both within-$w$ and cross-$W$ clusters), as well as to roughly quarter of the zero counts associating elements with the rest of the features.

The plain IB method consistently produced configurations strongly correlated with the masking clustering, while the CP algorithm revealed the target configuration. We got (see Table 1A) almost perfect results in configurations of nearly equal-sized cross-$W$ clusters, and somewhat less perfect reconstruction in configurations of diverging sizes (6, 9, 15, 21 and 24). Performance level was measured relatively to optimal target-output cluster match by the proportion of elements correctly assigned, where assignment of an element $x$ follows its highest $p(c|x)$. The results indicated were averaged over 200 runs. They were obtained for the optimal $\eta$, which was found to be higher in the diverging-sizes task.

In the text-based task, the clustered elements – keywords – were automatically extracted from five distinct corpora addressing five religions: introductory web pages, online magazines, encyclopedic entries etc., all downloaded from the Internet. The clustered keyword set $X$ was consequently pre-partitioned to disjoint subsets $\{X_w\}_{w \in W}$, one for each religion[4] ($|X_w| \approx 200$ for each $w$). We conducted experiments simultaneously involving religion pairs as well as all five religions. We took the features $Y$ to be a set of words that commonly occur within all five corpora ($|Y| \approx 7000$). $x$–$y$ co-occurrences were recorded within ±5-word sliding window truncated by sentence boundaries. $\eta$ was fixed to a value (1.0) enabling the formation of 20 clusters in all settings. The obtained clusters revealed interesting cross religion themes (see Sec. 1). For instance, the cluster (one of nine) capturing the theme of sacred festivals: the three highest $p(c/x)$ members within each religion were *Full-moon*, *Ceremony*, *Celebration* (Buddhism); *Easter*, *Sunday*, *Christmas*

Table 1: Average correct assignment proportion scores for the synthetic task (A) and Jaccard-coefficient scores for the religion keyword classification task (B).

| A. Synthetic Data | IB | CP |
| --- | --- | --- |
| equal-size clusters | .305 | .985 |
| non-equal clusters | .292 | .827 |

| B. Religion Data | IB | Coupled Clustering [7] | CP |
| --- | --- | --- | --- |
| (cross-expert agreement on religion pairs .462±.232) | | | |
| religion pairs | .200±.100 | .220±.138 | .407±.144 |
| all five (one case) | .104 | ——— | .167 |

(Chrsitianity); *Puja*, *Ceremony*, *Festival* (Hinduism); *Id-al-Fitr*, *Friday*, *Ramadan*, (Islam); and *Sukkoth*, *Shavuot*, *Rosh-Hodesh* (Judaism). The closest cluster produced by the plain IB method was poorer by far, including Islamic *Ramadan*, and *Id* and Jewish *Passover*, *Rosh-Hashanah* and *Sabbath* (which our method ranked high too), but no single related term from the other religions.

Our external evaluation standards were cross-religion keyword classes constructed manually by experts of comparative religion studies. One such expert classification involved all five religions, and eight classifications addressed religions in pairs. Each of the eight religion-pair classifications was contributed by two independent experts using the same keywords, so we could also assess the agreement between experts. As an overlap measure we employed the Jaccard coefficient: the number of element pairs co-assigned together by both one of the evaluated clusters and one of the expert classes, divided by the number of pairs co-assigned by either our clusters or the expert (or both). We did not assume the number of expert classes is known in advance (as done in the synthetic experiments), so the results were averaged over all configurations of 2–16 cluster hierarchy, for each experiment. The results shown in Table 1B – clear improvement relatively to plain IB and the distance-based coupled clustering [7] – are, however, persistent when the number of clusters is taken to be equal to the number of classes, or if only the best score in hierarchy is considered. The level of cross-expert agreement indicates that our results are reasonably close to the scores expected in such subjective task.

## 5    Comparison to Related Work

The information bottleneck framework served as the basis for several approaches that represent additional information in their clustering setting. The multivariate information bottleneck (MIB) adapts the IB framework for networks of multiple variables [3]. However, all variables in such networks are either compressed (like $X$), or predicted (like $Y$). The incorporation of an empirical variable to be masked or defocused in the sense of our $W$ is not possible. Including such variables in the MIB framework might be explored in future work.

Particularly relevant to our work is the IB-based method for extracting relevant constructs with *side information* [4]. This approach addresses settings in which two different types of features are distinguished explicitly: relevant versus irrelevant ones, denoted by $Y^+$ and $Y^-$. Both types of features are incorporated within a single functional to be minimized: $L^{(IB\text{-}side\text{-}info)} = I(C;X) - \beta(I(C;Y^+) - \gamma I(C;Y^-))$, which directly drives clustering to de-correlate $C$ and $Y^-$.

Formally, our setting can be mapped to the side information setting by regarding the pre-partition $W$ simply as the additional set of irrelevant features, giving symmetric (and opposite) roles to $W$ and $Y$. However, it seems that this view does not address properly the desired cross-partition setting. In our setting, it is assumed that clustering should be guided in general by $Y$, while $W$ should only neutralize particular information within $Y$ that would otherwise yield the undesired correlation between $C$ and $W$ (as described in Section 3.1). For that reason, the defocusing functional tie the three variables together by conditioning the de-correlation of $C$ and $W$ on $Y$, while its underlying assumption ensures the global de-correlation. Indeed, our method was found empirically superior on the cross-dataset task. The side-information IB method (the iterative algorithm with best scoring $\gamma$) achieves correct assignment proportion of 0.52 in both synthetic tasks, where our method scored 0.99 and 0.83 (see Table 1A) and, in the religion-pair keyword classification task, Jaccard coefficient improved by 20% relatively to plain IB (compared to our 100% improvement, see Table 1B).

# 6 Conclusions

This paper addressed the problem of clustering a pre-partitioned dataset, aiming to detect new internal structures that are not correlated with the pre-given partition but rather cut across its components. The proposed framework extends the cross-dataset clustering algorithm [8], providing better formal grounding and representing any pre-given (soft) partition of the dataset. Supported by empirical evidence, we suggest that our framework is better suited for the cross-partition task than applying the side-information framework [4], which was originally developed to address a somewhat different setting. We also demonstrate substantial empirical advantage over the distance-based coupled-clustering algorithm [7].

As an applied real-world goal, the algorithm successfully detects cross-religion commonalities. This goal exemplifies the more general notion of detecting analogies across different systems, which is a somewhat vague and non-consensual task and therefore especially challenging for a computational framework. Our approach can be viewed as an initial step towards principled identification of "hidden" commonalities between substantially different real world systems, while suppressing the vast majority of attributes that are irrelevant for the analogy.

Further research may study the role of defocusing in supervised learning, where some pre-given partitions might mask the role of underlying discriminative features. Additionally, it would be interesting to explore relationships to other disciplines, e.g., network information theory ([9], Ch. 14) which provided motivation for the side-information approach. Finally, both frameworks (ours and side-information) suggest the importance of dealing wisely with information that should *not* dictate the clustering output directly.

## Acknowledgments

We thank Yuval Krymolowski for helpful discussions and Tiina Mahlamäki, Eitan Reich and William Shepard, for contributing the religion keyword classifications.

## Footnotes

[1] Notice that "*Z* explaining well the dependencies between *A* and *B*" is equivalent with "*A* and *B* sharing little information in common given *Z*", i.e. low $I(A;B|Z)$. Complete conditional independence is exemplified in the IB framework, assuming $I(C;Y|X) = 0$.

[2] Eq. (4) resembles our suggestion in [8] to compute a geometric average over the subsets; in the current paper this scheme is analytically derived from the ID functional.

[3] High $\eta$ values tend to dictate centroids with features that are unevenly distributed across $W$, resulting in shrinkage of some of the clusters. Further analysis will be provided in future work.

[4] A keyword $x$ that appeared in the corpora of different religions was considered as a distinct element for each religion, so the $X_w$ were kept disjointed.

## References

[1] Hofmann, T. (2001) Unsupervised learning by probabilistic latent semantic analysis. *Journal of Machine Learning Research*, 41(1):177-196.

[2] Wagstaff K., Cardie C., Rogers S. and Schroedl S., 2001. Constrained K-Means clustering with background knowledge. *The 18th International Conference on Machine Learning (ICML-2001)*, pp 577-584.

[3] Friedman N., Mosenzon O., Slonim N. & Tishby N. (2002) Multivariate information bottleneck. *The 17th conference on Uncertainty in Artificial Intelligence (UAI-17)*, pp. 152-161.

[4] Chechik G. & Tishby N. (2002) Extracting relevant structures with side information. *Advances in Neural Processing Information Systems 15* (*NIPS'02*).

[5] Globerson, A., Chechik G. & Tishby N. (2003) Sufficient dimensionality reduction. *Journal of Machine Learning Research*, 3:1307-1331.

[6] Tishby, N., Pereira, F. C. & Bialek, W. (1999) The information bottleneck method. *The 37th Annual Allerton Conference on Communication, Control, and Computing*, pp. 368-379.

[7] Marx, Z., Dagan, I., Buhmann, J. M. & Shamir E. (2002) Coupled clustering: A method for detecting structural correspondence. *Journal of Machine Learning Research*, 3:747-780.

[8] Dagan, I., Marx, Z. & Shamir E (2002) Cross-dataset clustering: Revealing corresponding themes across multiple corpora. *Proceedings of the 6th Conference on Natural Language Learning (CoNLL-2002)*, pp. 15-21.

[9] Cover T. M. & Thomas J. A. (1991) *Elements of Information Theory*. John Wiley & Sons, Inc., New York, New York.
